# Early Brain Damage

**Volker Tresp, Ralph Neuneier and Hans Georg Zimmermann***
Siemens AG, Corporate Technologies
Otto-Hahn-Ring 6
81730 München, Germany

## Abstract

*Optimal Brain Damage* (OBD) is a method for reducing the number of weights in a neural network. OBD estimates the increase in cost function if weights are pruned and is a valid approximation if the learning algorithm has converged into a local minimum. On the other hand it is often desirable to terminate the learning process before a local minimum is reached (early stopping). In this paper we show that OBD estimates the increase in cost function incorrectly if the network is not in a local minimum. We also show how OBD can be extended such that it can be used in connection with early stopping. We call this new approach *Early Brain Damage*, EBD. EBD also allows to revive already pruned weights. We demonstrate the improvements achieved by EBD using three publicly available data sets.

## 1    Introduction

*Optimal Brain Damage* (OBD) was introduced by Le Cun *et al.* (1990) as a method to significantly reduce the number of weights in a neural network. By reducing the number of free parameters, the variance in the prediction of the network is often reduced considerably which —in some cases— leads to an improvement in generalization performance of the neural network. OBD might be considered a realization of the principle of Occam's razor which states that the simplest explanation (of the training data) should be preferred to more complex explanations (requiring more weights).

If $E$ is the cost function which is minimized during training, OBD calculates the

---

{Volker.Tresp, Ralph.Neuneier, Georg.Zimmermann}@mchp.siemens.de.

saliency of each parameter $w_i$ defined as

$$OBD(w_i) = A(w_i) = \frac{1}{2}\frac{\partial^2 E}{\partial w_i^2}w_i^2.$$

Weights with a small $OBD(w_i)$ are candidates for removal. $OBD(w_i)$ has the intuitive meaning of being the increase in cost function if weight $w_i$ is set to zero under the assumptions

- that the cost function is quadratic,
- that the cost function is "diagonal" which means it can be written as $E = Bias + 1/2\sum_i h_i(w_i - w_i^*)^2$ where where $\{w_i^*\}_{i=1}^W$ are the weights in a (local) optimum of the cost function (Figure 1) and the $h_i$ and $BIAS$ are parameters which are dependent on the training data set.
- and that $w_i \approx w_i^*$.

In practice, all three assumptions are often violated but experiments have demonstrated that OBD is a useful method for weight removal.

In this paper we want to take a closer look at the third assumption, i. e. the assumption that weights are close to optimum. The motivation is that theory and practice have shown that it is often advantageous to perform *early stopping* which means that training is terminated before convergence. Early stopping can be thought of as a form of regularization: since training typically starts with small weights, with early stopping weights are biased towards small weights analogously to other regularization methods such as ridge regression and weight decay. According to the assumptions in OBD we might be able to apply OBD only in heavily overtrained networks where we loose the benefits of early stopping. In this paper we show that OBD can be extended such that it can work together with early stopping. We call the new criterion *Early Brain Damage* (EBD). As in OBD, EBD contains a number of simplifying assumptions which are typically invalid in practice. Therefore, experimental results have to demonstrate that EBD has benefits. We validate EBD using three publicly available data sets.

## 2  Theory

As in OBD we approximate the cost function locally by a quadratic function and assume a "diagonal" form. Figure 1 illustrates that $OBD(w_i)$ for $w_i = w_i^*$ calculates the increase in cost function if $w_i$ is set to zero. In early stopping where $w_i \neq w_i^*$, $OBD(w_i)$ calculates the quantity denoted as $A_i$ in Figure 1. Consider

$$B_i = -\frac{\partial E}{\partial w_i}w_i.$$

The saliency of weights $w_i$ in *Early Stopping Pruning*

$$ESP(w_i) = A_i + B_i$$

is an estimate of how much the cost function increases if the *current* $w_i$ (i. e. $w_i$ in early stopping) is set to zero. Finally, consider

$$C_i = \frac{1}{2}\left(\frac{\partial^2 E}{\partial w_i^2}\right)^{-1}\left(\frac{\partial E}{\partial w_i}\right)^2.$$

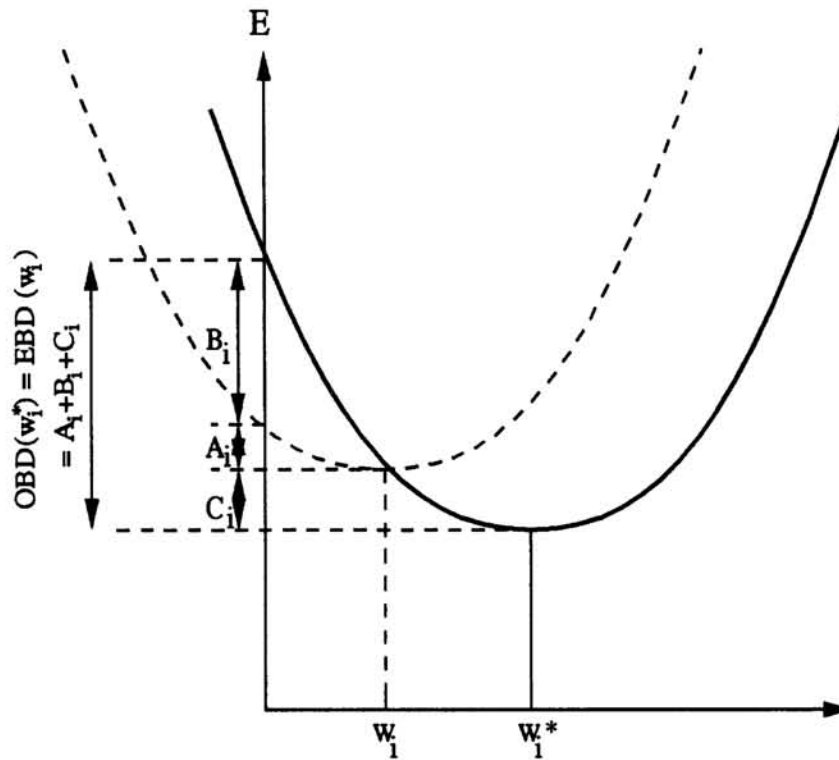

Figure 1: The figure shows the cost function $E$ as a function of one weight $w_i$ in the network. $w_i^*$ is the optimal weight. $w_i$ is the weight at an early stopping point. If OBD is applied at $w_i$, it estimates the quantity $A_i$. $ESP(w_i) = A_i + B_i = E(w_i) - E(w_i = 0)$ estimate the increase in cost function if $w_i$ is pruned. $EBD(w_i) = A_i + B_i + C_i = E(w_i^*) - E(w_i = 0)$ is the difference in cost function if we would train to convergence and if we would set $w_i = 0$. In other words $EBD(w_i) = OBD(w_i^*)$.

The saliency of weight $w_i$ in EBD is

$$EBD(w_i) = OBD(w_i^*) = A_i + B_i + C_i$$

which estimates the increase in cost function if $w_i$ is pruned after convergence (i.e. $EBD(w_i) = OBD(w_i^*)$) but based on local information around the current value of $w_i$. In this sense EBD evaluates the "potential" of $w_i$. Weights with a small $EBD(w_i)$ are candidates for pruning.

Note, that all terms required for EBD are easily calculated. With a quadratic cost function $E = \sum_{k=1}^{K}(y^k - NN(x^k))^2$ OBD approximates (OBD-approximation)

$$\frac{\partial^2 E}{\partial w_i^2} \approx 2 \sum_{k=1}^{K} \left( \frac{\partial NN(x^k)}{\partial w_i} \right)^2 \tag{1}$$

where $(x^k, y^k)_{k=1}^{K}$ are the training data and $NN(x^k)$ is the network response.

## 3 Extensions

### 3.1 Revival of Weights

In some cases, it is beneficial to revive weights which are already pruned. Note, that $C_i$ exactly estimate the decrease in cost function if weight $w_i$ is "revived". Weights with a large $C_i(w_i = 0)$ are candidates for revival.

### 3.2 Early Brain Surgeon (EBS)

After OBD or EBD is performed, the network needs to be retrained since the "diagonal" approximation is typically violated and there are dependencies between weights. *Optimal Brain Surgeon* (OBS, Hassibi and Storck, 1993) does not use the "diagonal" approximation and recalculates the new weights without explicit retraining. OBS still assumes a quadratic approximation of the cost function. The saliency in OBS is

$$L_i = \frac{w_i^2}{2[H^{-1}]_{ii}}$$

where $[H^{-1}]_{ii}$ is $i$-th diagonal element of the inverse of the Hessian. $L_i$ estimates the increase in cost if the $i$-th weight is set to zero and all other weights are retrained. To recalculate all weights after weight $w_i$ is removed apply

$$w_{new} = w_{old} - \frac{w_i}{[H^{-1}]_{ii}} H^{-1} e_i$$

where $e_i$ is the unit vector in the $i$-th direction.

Analogously to OBS, *Early Brain Surgeon* EBS would first calculate the optimal weight vector using a second order approximation of the cost function

$$\hat{w}^* = w - H^{-1} \frac{\partial E}{\partial w}$$

and then apply OBS using $\hat{w}^*$. We did not pursue this idea any further since our initial experiments indicated that $w^*$ was not estimated very accurately in praxis. Hassibi *et al.* (1994) achieved good performance with OBS even when weights were far from optimal.

## 3.3 Approximations to the Hessian and the Gradient

Finnoff *et al.* (1993) have introduced the interesting idea that the relevant quantities for OBD can be estimated from the statistics of the weight changes.

Consider the update in pattern by pattern gradient descent learning and a quadratic cost function

$$\Delta w_i = -\eta \frac{\partial E_k}{\partial w} = 2\eta(y^k - NN(x^k))\frac{\partial NN(x^k)}{\partial w_i}$$

with $E_k = (y_k - NN(x_k))^2$ where $\eta$ is the learning rate.

We assume that $x^k$ and $y^k$ are drawn online from a fixed distribution (which is strictly not true since in pattern by pattern learning we draw samples from a fixed training data set). Then, using the quadratic and "diagonal" approximation of the cost function and assuming that the noise $\epsilon$ in the model

$$y^k = NN(x^k) + \epsilon^k$$

is additive uncorrelated with variance $\sigma^2$ [1]

$$\mathcal{E}(\Delta w_i) \approx \frac{1}{K}\eta\frac{\partial E}{\partial w} \tag{2}$$

and

$$VAR(\Delta w_i) = VAR\left(2\eta(y^k - NN(x^k))\frac{\partial NN(x^k)}{\partial w_i}\right)$$

$$= 4\eta^2 VAR\left((y^k - NN^*(x^k))\frac{\partial NN(x^k)}{\partial w_i}\right) + 4\eta^2 VAR\left((w_i^* - w_i)\left(\frac{\partial NN(x^k)}{\partial w_i}\right)^2\right)$$

$$= 4\sigma^2\eta^2\mathcal{E}\left(\frac{\partial NN(x^k)}{\partial w_i}\right)^2 + 4\eta^2(w_i^* - w_i)^2 VAR\left(\left(\frac{\partial NN(x^k)}{\partial w_i}\right)^2\right)$$

where $NN^*(x^k)$ is the network output with optimal weights $\{w_i^*\}_{i=1}^W$. Note, that in the OBD approximation (Equation 1)

$$\mathcal{E}\left(\frac{\partial NN(x^k)}{\partial w_i}\right)^2 \approx \frac{1}{2K}\frac{\partial^2 E}{\partial w_i^2}$$

and

$$w - w* \approx \frac{1/\eta\,\mathcal{E}(\Delta w_i)}{2\,\mathcal{E}\left(\frac{\partial NN(x^k)}{\partial w_i}\right)^2}.$$

If we make the further assumption that $\partial NN(x^k)/\partial w_i$ is Gaussian distributed with zero mean[2]

$$VAR\left(\left(\frac{\partial NN(x^k)}{\partial w_i}\right)^2\right) = 2\left(\mathcal{E}\left(\frac{\partial NN(x^k)}{\partial w_i}\right)^2\right)^2$$

we obtain

$$VAR(\Delta w_i) = \frac{2}{K}\sigma^2\eta^2\frac{\partial^2 E}{\partial w_i^2} + 2 * (\mathcal{E}(\Delta w_i))^2. \qquad (3)$$

The first term in Equation 3 is a result of the residual error which is translated into weight fluctuations. But note, that weights with a small variance with a large $\partial^2 E/\partial w_i^2$ fluctuate the most. The first term is only active when there is a residual error, i.e. $\sigma^2 > 0$. The second term is non-zero independent of $\sigma^2$ and is due to the fact that in sample-by-sample learning, weight updates have a random component. From Equation 2 and Equation 3 all terms needed in EBD (i. e. $\partial E/\partial w$, and $\partial^2 E/\partial w_i^2$) are easily estimated.

## 4  Experimental Results

In our experiments we studied the performance of OBD, ESP and EBD in connection with early stopping. Although theory tells us that EBD should provide the best estimate of the the increase in cost function by the removal of weight $w_i$, it is not obvious how reliable that estimate is when the assumptions ("diagonal" quadratic cost function) are violated. Also we are not really interested in the correct estimate of the increase in cost function but in a ranking of the weights. Since the assumptions which go into OBD, EBD, ESP (and also OBS and EBS) are questionable, the usefulness of the new methods have to be demonstrated using practical experiments.

We used three different data sets: Breast Cancer Data, Diabetes Data, and Boston Housing Data. All three data sets can be obtained from the UCI repository (ftp:ics.uci.edu/pub/machine-learning-databases). The Breast Cancer Data contains 699 samples with 9 input variables consisting of cellular characteristics and one binary output with 458 benign and 241 malignant cases. The Diabetes Data contains 768 samples with 8 input variables and one binary output. The Boston Housing Data consist of 506 samples with 13 input variables which potentially influence the housing price (output variable) in a Boston neighborhood (Harrison & Rubinfeld, 1978).

Our procedure is as follows. The data set is divided into training data, validation data and test data. A neural network (MLP) is trained until the error on the validation data set starts to increase. At this point OBD, ESP and EBD are employed and 50% of all weights are removed. After pruning the networks are retrained until again the error on the validation set starts to increase. At this point the results are compared. Each experiment was repeated 5-times with different divisions of the data into training data, validation data and test data and we report averages over those 5 experiments.

Table 1 sums up the results. The first row shows the number of data in training set, validation set and test set. The second row displays the test set error at the (first) early stopping point. Rows 3 to 5 show test set performance of OBD, ESP and EBD at the stopping point after pruning and retraining (absolute / relative to early stopping). In all three experiments, EBD performed best and OBD was second best in two experiments (Breast Cancer Data and Diabetes Data). In two experiments (Breast Cancer Data and Boston Housing Data) the performance after pruning improved.

Table 1: Comparing OBD, ESP, and EBD.

|              | Breast Cancer    | Diabetes        | Boston Housing Data |
|--------------|------------------|-----------------|---------------------|
| Train/V/Test | 233/233/233      | 256/256/256     | 168/169/169         |
| Hidden units | 10               | 5               | 3                   |
| MSE (Stopp)  | 0.0340           | 0.1625          | 0.2283              |
| OBD          | 0.0328 / 0.965   | 0.1652 / 1.017  | 0.2275 /0.997       |
| ESP          | 0.0331 / 0.973   | 0.1657 /1.020   | 0.2178 / 0.954      |
| EBD          | 0.0326 / 0.959   | 0.1647 /1.014   | 0.2160 / 0.946      |

## 5  Conclusions

In our experiments, EBD showed better performance than OBD if used in conjunction with early stopping. The improvement in performance is not dramatic which indicates that the rankings of the weights in OBD are reasonable as well.

## Footnotes

[1] $\mathcal{E}$ stands for the expected value. With $w_i$ kept at a fixed value.

[2] The zero mean assumption is typically violated but might be enforced by renormalization.

## References

Finnoff, W., Hergert, F., and Zimmermann, H. (1993). Improving model selection by nonconvergent methods, *Neural Networks*, Vol. 6, No. 6.

Hassibi, B. and Storck, D. G. (1993). Second order derivatives for network pruning: Optimal Brain Surgeon. In: Hanson, S. J., Cowan, J. D., and Giles, C. L. (Eds.). *Advances in Neural Information Processing Systems 5*, San Mateo, CA, Morgan Kaufman.

Hassibi, B., Storck, D. G., and Wolff, G. (1994). Optimal Brain Surgeon: Extensions and performance comparisons. In: Cowan, J. D., Tesauro, G., and Alspector, J. (Eds.). *Advances in Neural Information Processing Systems 6*, San Mateo, CA, Morgan Kaufman.

Le Cun, Y., Denker, J. S., and Solla, S. A. (1990). Optimal brain damage. In: D. S. Touretzky (Ed.). *Advances in Neural Information Processing Systems 2*, San Mateo, CA, Morgan Kaufman.